# Dynamics of Training

**Siegfried Bös**[*]
Lab for Information Representation
RIKEN, Hirosawa 2–1, Wako–shi
Saitama 351–01, Japan

**Manfred Opper**
Theoretical Physics III
University of Würzburg
97074 Würzburg, Germany

## Abstract

A new method to calculate the full training process of a neural network is introduced. No sophisticated methods like the replica trick are used. The results are directly related to the actual number of training steps. Some results are presented here, like the maximal learning rate, an exact description of early stopping, and the necessary number of training steps. Further problems can be addressed with this approach.

## 1 INTRODUCTION

Training guided by empirical risk minimization does not always minimize the expected risk. This phenomenon is called overfitting and is one of the major problems in neural network learning. In a previous work [Bös 1995] we developed an approximate description of the training process using statistical mechanics. To solve this problem exactly, we introduce a new description which is directly dependent on the actual training steps. As a first result we get analytical curves for empirical risk and expected risk as functions of the training time, like the ones shown in Fig. 1.

To make the method tractable we restrict ourselves to a quite simple neural network model, which nevertheless demonstrates some typical behavior of neural nets. The model is a single layer perceptron, which has one $N$–dim. layer of adjustable weights $\vec{W}$ between input $\vec{x}$ and output $z$. The outputs are linear, i.e.

$$z = h = \frac{1}{\sqrt{N}} \sum_{i=1}^{N} W_i x_i \,. \tag{1}$$

We are interested in supervised learning, where examples $x_i^\mu$ ($\mu = 1, ..., P$) are given for which the correct output $z_*$ is known. To define the task more clearly and to monitor the training process, we assume that the examples are provided by another network, the so called *teacher* network. The teacher is not restricted to linear outputs, it can have a nonlinear output function $g_*(h_*)$.

---

[*]email: boes@zoo.riken.go.jp and opper@physik.uni-wuerzburg.de

Learning by examples attempts to minimize the error averaged over all examples, i.e. $E_T := 1/2 < (z_*^\mu - z^\mu)^2 >_{\{\vec{x}^\mu\}}$, which is called *training error* or empirical risk. In fact what we are interested in is a minimal error averaged over all possible inputs $\vec{x}$, i.e $E_G := 1/2 < (z_* - z)^2 >_{\{\vec{x}\in\text{Input}\}}$, called *generalization error* or expected risk.

It can be shown [see Bös 1995] that for random inputs, i.e. all components $x_i$ are independent and have zero means and unit variance, the generalization error can be described by the order parameters $R$ and $Q$,

$$E_G(t) = \frac{1}{2}\left[G - 2H\,R(t) + Q(t)\right] \quad \text{with} < \ldots >_h = \int\limits_{-\infty}^{\infty} \frac{dh}{\sqrt{2\pi}}\, e^{-\frac{h^2}{2}} \ldots, \quad (2)$$

with the two parameters $G = <\left[g_*(h)\right]^2>_h$ and $H = <g_*(h)\,h>_h$. The order parameters are defined as:

$$R(t) = < \frac{1}{N}\sum_{i=1}^{N} W_i^* W_i(t) >_{\{W_i^*\}}, \quad Q(t) = < \frac{1}{N}\sum_{i=1}^{N}(W_i(t))^2 >_{\{W_i^*\}}. \quad (3)$$

As a novelty in this paper we average the order parameters not as usual in statistical mechanics over many example realizations $\{x_i^\mu\}$, but over many teacher realizations $\{W_i^*\}$, where we use a spherical distribution. This corresponds to a Bayesian average over the unknown teacher. A study of the static properties of this model was done by Saad [1996]. Further comments about the averages can be found in the appendix.

In the next section we introduce our new method briefly. Readers, who do not wish to go into technical details in first reading, can turn directly to the results (15) and (16). The remainder of the section can be read later, as a proof. In the third section results will be presented and discussed. Finally, we conclude the paper with a summary and a perspective on further problems.

## 2 DYNAMICAL APPROACH

Basically we exploit the gradient descent learning rule, using the linear student, i.e $g'(h) = 1$ and $z^\mu = h^\mu = \frac{1}{\sqrt{N}}\vec{W}\vec{x}^\mu$,

$$W_i(t+1) = W_i(t) - \eta\frac{\partial(P E_T)}{\partial W_i} = W_i(t) + \frac{\eta}{\sqrt{N}}\sum_{\mu=1}^{P}(z_*^\mu - z^\mu)x_i^\mu, \quad (4)$$

For $P < N$, the weights are linear combinations of the example inputs $x_i^\mu$, if $W_i(0) = 0$,

$$W_i(t) =: \frac{1}{\sqrt{N}}\sum_{\mu=1}^{P}\sigma_\mu(t)\,x_i^\mu. \quad (5)$$

After some algebra a recursion for $\sigma_\mu(t)$ can be found, i.e.

$$\sigma_\mu(t+1) = \sum_{\nu=1}^{P}\left[\delta_{\mu\nu} - \eta\left(\frac{1}{N}\sum_{i=1}^{N}x_i^\mu x_i^\nu\right)\right]\sigma_\nu(t) + \eta z_*^\mu, \quad (6)$$

where the term in the round brackets defines the *overlap matrix* $\mathbf{C}_{\mu\nu}$. From the geometric series we know the solution of this recursion, and therefore for the weights

$$W_i(t) = \frac{\eta}{\sqrt{N}}\sum_{\mu,\nu=1}^{P} z_*^\mu \left[\frac{\mathbf{E} - (\mathbf{E} - \eta\,\mathbf{C})^t}{\mathbf{E} - (\mathbf{E} - \eta\,\mathbf{C})}\right]_{\mu\nu} x_i^\nu. \quad (7)$$

It fulfills the initial conditions $W_i(0) = 0$ and $W_i(1) = \eta \sum_{\mu=1}^{P} z_*^\mu x_i^\mu$ (Hebbian), and yields after infinite time steps the so called *Pseudo-inverse* weights, i.e.

$$W_i(\infty) = \frac{1}{\sqrt{N}} \sum_{\mu,\nu=1}^{P} z_*^\mu \, (\mathbf{C}^{-1})_{\mu\nu} \, x_i^\nu \,. \qquad (8)$$

This is valid as long as the examples are linearly independent, i.e. $P < N$. Remarks about the other case $(P > N)$ will follow later.

With the expression (7) we can calculate the behavior of the order parameters for the whole training process. For $R(t)$ we get

$$
\begin{aligned}
R(t) &= \frac{1}{N} \sum_{\mu,\nu=1}^{P} \left[ \frac{\mathbf{E} - (\mathbf{E} - \eta\mathbf{C})^t}{\mathbf{C}} \right]_{\mu\nu} < z_*^\mu \left( \frac{1}{\sqrt{N}} \sum_{i=1}^{N} W_i^* x_i^\nu \right) >_{\{W_i^*\}} \\
&= \alpha H \left( 1 - \frac{1}{P} \sum_{\mu=1}^{P} \left[ (\mathbf{E} - \eta\mathbf{C})^t \right]_{\mu\mu} \right) .
\end{aligned} \qquad (9)
$$

For the average we have used expression (21) from the appendix. Similarly we get for the other order parameter,

$$
\begin{aligned}
Q(t) &= \frac{1}{N} \sum_{\mu,\nu,\tau,\sigma=1}^{P} \left[ \frac{\mathbf{E} - (\mathbf{E} - \eta\mathbf{C})^t}{\mathbf{C}} \right]_{\mu\nu} \left[ \frac{\mathbf{E} - (\mathbf{E} - \eta\mathbf{C})^t}{\mathbf{C}} \right]_{\tau\sigma} \\
&\quad \times < z_*^\mu z_*^\tau \left( \frac{1}{N} \sum_{i=1}^{N} x_i^\nu x_i^\sigma \right) >_{\{W_i^*\}} \\
&= \frac{\alpha(G - H^2)}{P} \sum_{\mu=1}^{P} \left[ \mathbf{C}^{-1} \left( \mathbf{E} - (\mathbf{E} - \eta\mathbf{C})^t \right)^2 \right]_{\mu\mu} \\
&\quad + \frac{\alpha H^2}{P} \sum_{\mu=1}^{P} \left[ \left( \mathbf{E} - (\mathbf{E} - \eta\mathbf{C})^t \right)^2 \right]_{\mu\mu} .
\end{aligned} \qquad (10)
$$

Again we have applied an identity (20) from the appendix and we did some matrix algebra. Note, up to this point the order parameters were calculated without any assumption about the statistics of the inputs. The results hold, even without the thermodynamic limit.

The trace can be calculated by an integration over the eigenvalues, thus we attain integrals of the following form,

$$\frac{1}{P} \sum_{\mu=1}^{P} \left[ (\mathbf{E} - \eta\mathbf{C})^l \, \mathbf{C}^m \right]_{\mu\mu} = \int_{\xi_{\min}}^{\xi_{\max}} d\xi \, \rho(\xi) \, (1 - \eta\xi)^l \, \xi^m =: I_m^l(t,\alpha,\eta) \,, \qquad (11)$$

with $l = \{0, t, 2t\}$ and $m = \{-1, 0, 1\}$.

These integrals can be calculated once we know the density of the eigenvalues $\rho(\xi)$. The determination of this density can be found in recent literature calculated by Opper [1989] using replicas, by Krogh [1992] using perturbation theory and by Sollich [1994] with matrix identities. We should note, that the thermodynamic limit and the special assumptions about the inputs enter the calculation here. All authors found

$$\rho(\xi) = \frac{1}{2\pi\alpha\xi} \sqrt{(\xi_{\max} - \xi)(\xi - \xi_{\min})} \,, \qquad (12)$$

for $\alpha < 1$. The maximal and the minimal eigenvalues are $\xi_{\max,\min} := (1 \pm \sqrt{\alpha})^2$. So all that remains now is a numerical integration.

Similarly we can calculate the behavior of the training error from

$$E_{\mathrm{T}}(t) = < \frac{1}{2P} \sum_{\mu=1}^{P} (z_*^\mu - h^\mu)^2 >_{\{w_i^*\}} .$$  (13)

For the overdetermined case ($P > N$) we can find a recursion analog to (6),

$$W_i(t+1) = \sum_{j=1}^{N} \left[ \delta_{ij} - \left( \frac{1}{N} \sum_{\mu=1}^{P} x_i^\mu x_j^\mu \right) \right] W_j(t) + \frac{\eta}{\sqrt{N}} \sum_{\mu=1}^{P} z_*^\mu x_i^\mu .$$  (14)

The term in the round brackets defines now the matrix $\mathbf{B}_{ij}$. The calculation is therefore quite similar to above with the matrix $\mathbf{B}$ playing the role of matrix $\mathbf{C}$. The density of the eigenvalues $\rho(\lambda)$ for matrix $\mathbf{B}$ is the one from above (12) multiplied by $\alpha$.

Altogether, we find the following results in the case of $\alpha < 1$,

$$E_{\mathrm{G}}(t,\alpha,\eta) = \frac{G}{2} + \frac{G - H^2}{2} \alpha \left( \frac{1}{1-\alpha} - 2I_{-1}^t + I_{-1}^{2t} \right) - \frac{H^2}{2} \alpha \left( 1 - I_0^{2t} \right) ,$$

$$E_{\mathrm{T}}(t,\alpha,\eta) = \frac{G - H^2}{2} I_0^{2t} + \frac{H^2}{2} I_1^{2t} ,$$  (15)

and in the case of $\alpha > 1$,

$$E_{\mathrm{G}}(t,\alpha,\eta) = \frac{G - H^2}{2} \left( 1 + \frac{1}{\alpha - 1} - 2I_{-1}^t + I_{-1}^{2t} \right) + \frac{H^2}{2} I_0^{2t} ,$$

$$E_{\mathrm{T}}(t,\alpha,\eta) = \frac{G - H^2}{2} \left( 1 - \frac{1}{\alpha} + \frac{I_0^{2t}}{\alpha} \right) + \frac{H^2}{2} \frac{I_1^{2t}}{\alpha} .$$  (16)

If $t \to \infty$ all the time–dependent integrals $I_k^t$ and $I_k^{2t}$ vanish. The remaining first two terms describe, in the limit $\alpha \to \infty$, the optimal convergence rate of the errors. In the next section we discuss the implications of this result.

## 3  RESULTS

First we illustrate how well the theoretical results describe the training process. If we compare the theory with simulations, we find a very good correspondence, see Fig. 1.

Trying other values for the learning rate we can see that there is a maximal learning rate. It is twice the inverse of the maximal eigenvalue of the matrix $\mathbf{B}$, i.e.

$$\eta_{\max} = \frac{2}{\xi_{\max}} = \frac{2}{(1 + \sqrt{\alpha})^2} .$$  (17)

This is consistent with a more general result, that the maximal learning is twice the inverse of the maximal eigenvalue of the Hessian. In the case of the linear perceptron the matrix $\mathbf{B}$ is identical to the Hessian.

As our approach is directly related to the actual number of training steps we can examine how training time varies in different training scenarios. Training can be stopped if the training error reaches a certain minimal value, i.e if $E_{\mathrm{T}}(t) \leq E_{\mathrm{T}}^{\min} + \epsilon$. Or, in cross-validated early stopping, we will terminate training if the generalization error starts to increase, i.e. if $E_{\mathrm{G}}(t+1) > E_{\mathrm{G}}(t)$.

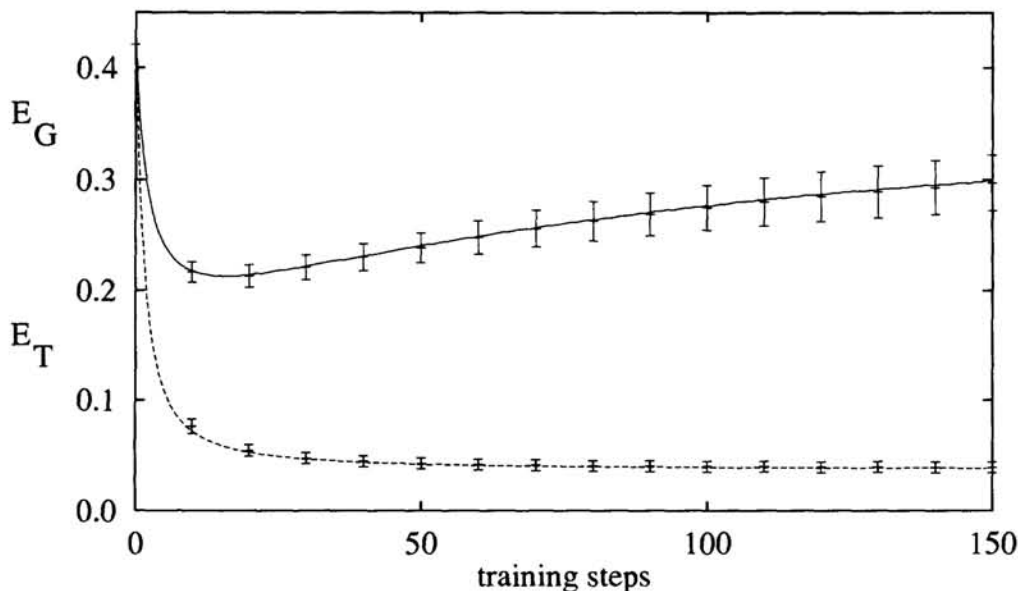

Figure 1: Behavior of the generalization error $E_G$ (upper line) and the training error $E_T$ (lower line) during the training process. As the loading rate $\alpha = P/N = 1.5$ is near the storage capacity ($\alpha = 1$) of the net overfitting occurs. The theory describes the results of the simulations very well. Parameters: learning rate $\eta = 0.1$, system size $N = 200$, and $g_*(h) = \tanh(\gamma h)$ with gain $\gamma = 5$.

Fig. 2 shows that in exhaustive training the training time diverges for $\alpha$ near 1, in the region where also the overfitting occurs. In the same region early stopping shows only a slight increase in training time.

Furthermore, we can guess from Fig. 2 that asymptotically only a few training steps are necessary to fulfill the stopping criteria. This has to be specified more precisely. First we study the behavior of $E_G$ after only one training step, i.e. $t = 1$. Since we interested in the limit of many examples ($\alpha \to \infty$) we choose the learning rate as a fraction of the maximal learning rate (17), i.e. $\eta = \eta_0/\xi_{\max}$. Then we can calculate the behavior of $E_G(t = 1, \alpha, \xi_{\max}^{-1})$ analytically. We find that only in the case of $\eta_0 = 1$, the generalization error can reach its asymptotic minimum $E_{\inf}$. The rate of the convergence is $\alpha^{-1}$ like in the optimal case, but the prefactor is different.

However, already for $t = 2$ we find,

$$\bar{E}_G\left(t = 2, \alpha, \eta = \xi_{\max}^{-1}\right) := E_G - E_{\inf} = \frac{G - H^2}{2} \frac{1}{\alpha - 1} + \mathcal{O}\left(\frac{1}{\alpha^2}\right). \quad (18)$$

If $\alpha$ is large, so that we can neglect the $\alpha^{-2}$ term, then two batch training steps are already enough to get the optimal convergence rate. These results are illustrated in Fig. 3.

## 4 SUMMARY

In this paper we have calculated the behavior of the learning and the training error during the whole training process. The novel approach relates the errors directly to the actual number of training steps. It was shown how good this theory describes the training process. Several results have been presented, such as the maximal learning rate and the training time in different scenarios, like early stopping. If the learning rate is chosen appropriately, then only two batch training steps are necessary to reach the optimal convergence rate for sufficiently large $\alpha$.

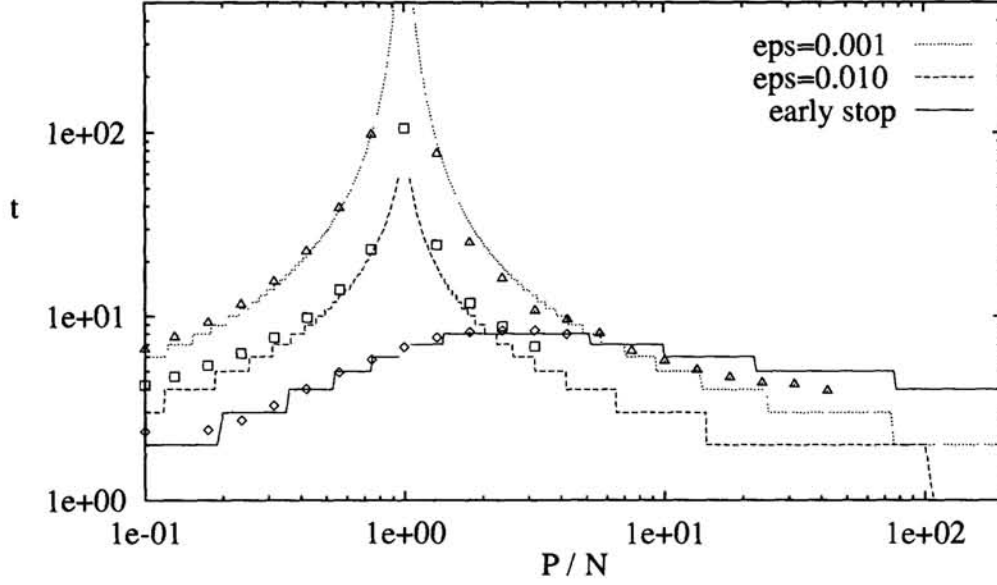

Figure 2: Number of necessary training steps to fulfill certain stopping criteria. The upper lines show the result if training is stopped when the training error is lower than $E_{\mathrm{T}}^{\min}+\epsilon$, with $\epsilon = 0.001$ (dotted line) and $\epsilon = 0.01$ (dashed line). The solid line is the early stopping result where training is stopped, when the generalization error started to increase, $E_{\mathrm{G}}(t+1) > E_{\mathrm{G}}(t)$. Simulation results are indicated by marks. Parameters: learning rate $\eta = 0.01$, system size $N = 200$, and $g_*(h) = \tanh(\gamma h)$ with gain $\gamma = 5$.

Further problems, like the dynamical description of weight decay, and the relation of the dynamical approach to the thermodynamic description of the training process [see Bös, 1995] can not be discussed here due to lack of space. These problems are examined in an extended version of this work [Bös and Opper 1996]. It would be very interesting if this method could be extended towards other, more realistic models.

## A   APPENDIX

Here we add some identities which are necessary for the averages over the teacher weight distributions, eqs. (9) and (10). In the statistical mechanics approach one assumes that the distribution of the local fields $h$ is Gaussian. This becomes true, if one averages over random inputs $x_i$, with first moments zero and one, which is the usual approach [see Bös 1995 and ref.]. In principle it is also possible to average over many tasks, i.e many teacher realizations $\vec{W}^*$, which is done here. The Gaussian local fields $h_*^\mu$ fulfill,

$$< h_*^\mu >= 0, \quad < h_*^\mu h_*^\nu >= C_{\mu\nu}. \tag{19}$$

This implies

$$< z_*^\mu z_*^\nu >_{\{W_i^*\}} = \int_{-\infty}^\infty D\tilde{h}_*^\mu \int_{-\infty}^\infty D\tilde{h}_*^\nu \, g_*(\sqrt{1-(C_{\mu\nu})^2}\,\tilde{h}_*^\mu + C_{\mu\nu}\,\tilde{h}_*^\nu)\, g_*(\tilde{h}_*^\nu)$$

$$= \delta_{\mu\nu}\, G + (C_{\mu\nu} - \delta_{\mu\nu})\, H^2. \tag{20}$$

In the second identity we first calculated the diagonal term and for the non-diagonal term we made an expansion assuming small correlations. Similarly the following

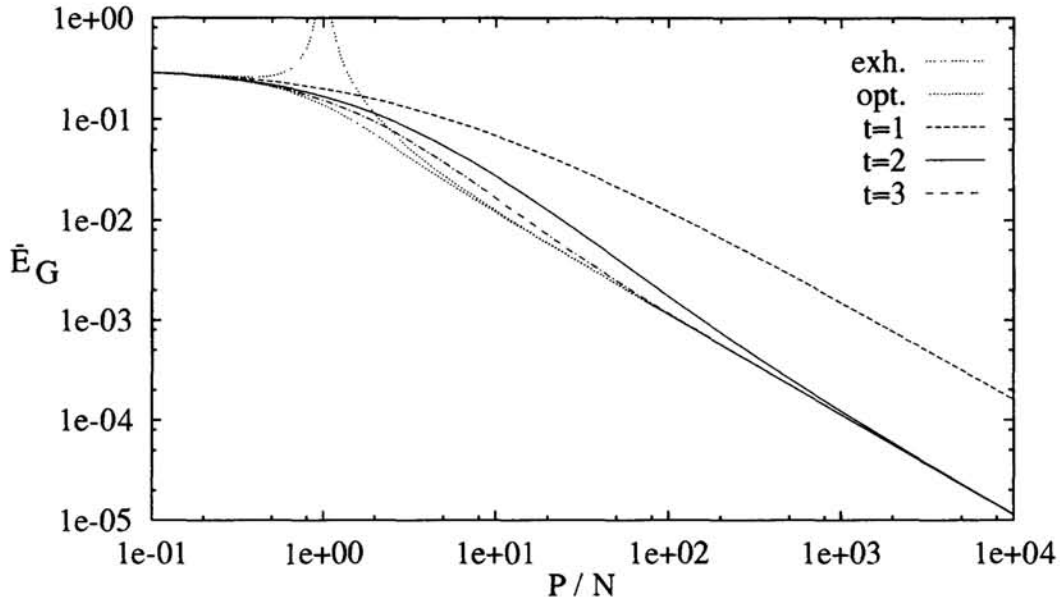

Figure 3: Behavior of $\bar{E}_G = E_G - E_{\text{inf}}$ after $t$ training steps. Results for $t = 1$, 2 and 3 are given. For large enough $\alpha$ it is already after $t = 2$ training steps possible to reach the optimal convergence (solid line). If $t = 3$ the optimal result is reached even faster. Parameters: learning rate $\eta = \xi_{\max}^{-1}$ and $g_*(h) = \tanh(\gamma h)$ with gain $\gamma = 5$.

identity can be proved,

$$< z_*^\mu h_*^\nu >_{\{W_i^*\}} = \delta_{\mu\nu} H + (C_{\mu\nu} - \delta_{\mu\nu}) H . \tag{21}$$

**Acknowledgment:** We thank Shun-ichi Amari for many discussions and E. Helle, A. Stevenin–Barbier for proofreading and valuable comments.

### References

Bös S. (1995), 'Avoiding overfitting by finite temperature learning and cross–validation', in *Int. Conference on Artificial Neural Networks 95 (ICANN'95)*, edited by EC2 & Cie, Vol.2, p.111–116.

Bös S., and Opper M. (1996), 'An exact description of early stopping and weight decay', submitted.

Kinzel W., and Opper M. (1995), 'Dynamics of learning', in *Models of Neural Networks I*, edited by E. Domany, J. L. van Hemmen and K. Schulten, Springer, p.157–179.

Krogh A. (1992), 'Learning with noise in a linear perceptron', *J. Phys. A* **25**, p.1135–1147.

Opper M. (1989), 'Learning in neural networks: Solvable dynamics', *Europhys. Lett.* **8**, p.389–392.

Saad D. (1996), 'General Gaussian priors for improved generalization', submitted to *Neural Networks*.

Sollich P. (1995), 'Learning in large linear perceptrons and why the thermodynamic limit is relevant to the real world', in *NIPS 7*, p.207–214.